# Kernels and learning curves for Gaussian process regression on random graphs

**Peter Sollich, Matthew J Urry**
King's College London, Department of Mathematics
London WC2R 2LS, U.K.
{peter.sollich,matthew.urry}@kcl.ac.uk

**Camille Coti**
INRIA Saclay Île de France, F-91893 Orsay, France

## Abstract

We investigate how well Gaussian process regression can learn functions defined on graphs, using large regular random graphs as a paradigmatic example. Random-walk based kernels are shown to have some non-trivial properties: within the standard approximation of a locally tree-like graph structure, the kernel does not become constant, i.e. neighbouring function values do not become fully correlated, when the lengthscale $\sigma$ of the kernel is made large. Instead the kernel attains a non-trivial limiting form, which we calculate. The fully correlated limit is reached only once loops become relevant, and we estimate where the crossover to this regime occurs. Our main subject are learning curves of Bayes error versus training set size. We show that these are qualitatively well predicted by a simple approximation using only the spectrum of a large tree as input, and generically scale with $n/V$, the number of training examples per vertex. We also explore how this behaviour changes for kernel lengthscales that are large enough for loops to become important.

## 1 Motivation and Outline

Gaussian processes (GPs) have become a standard part of the machine learning toolbox [1]. Learning curves are a convenient way of characterizing their capabilities: they give the generalization error $\epsilon$ as a function of the number of training examples $n$, averaged over all datasets of size $n$ under appropriate assumptions about the process generating the data. We focus here on the case of GP regression, where a real-valued output function $f(x)$ is to be learned. The general behaviour of GP learning curves is then relatively well understood for the scenario where the inputs $x$ come from a continuous space, typically $\mathbb{R}^n$ [2, 3, 4, 5, 6, 7, 8, 9, 10]. For large $n$, the learning curves then typically decay as a power law $\epsilon \propto n^{-\alpha}$ with an exponent $\alpha \leq 1$ that depends on the dimensionality $n$ of the space as well as the smoothness properties of the function $f(x)$ as encoded in the covariance function.

But there are many interesting application domains that involve *discrete* input spaces, where $x$ could be a string, an amino acid sequence (with $f(x)$ some measure of secondary structure or biological function), a research paper (with $f(x)$ related to impact), a web page (with $f(x)$ giving a score used to rank pages), etc. In many such situations, similarity between different inputs – which will govern our prior beliefs about how closely related the corresponding function values are – can be represented by edges in a *graph*. One would then like to know how well GP regression can work in such problem domains; see also [11] for a related online regression algorithm. We study this

problem here theoretically by focussing on the paradigmatic example of random regular graphs, where every node has the same connectivity.

Sec. 2 discusses the properties of random-walk inspired kernels [12] on such random graphs. These are analogous to the standard radial basis function kernels $\exp[-(x-x')^2/(2\sigma^2)]$, but we find that they have surprising properties on large graphs. In particular, while loops in large random graphs are long and can be neglected for many purposes, by approximating the graph structure as locally tree-like, here this leads to a non-trivial limiting form of the kernel for $\sigma \to \infty$ that is *not* constant. The fully correlated limit, where the kernel is constant, is obtained only because of the presence of loops, and we estimate when the crossover to this regime takes place.

In Sec. 3 we move on to the learning curves themselves. A simple approximation based on the graph eigenvalues, using only the known spectrum of a large tree as input, works well qualitatively and predicts the exact asymptotics for large numbers of training examples. When the kernel lengthscale is not too large, below the crossover discussed in Sec. 2 for the covariance kernel, the learning curves depend on the number of examples per vertex. We also explore how this behaviour changes as the kernel lengthscale is made larger. Sec. 4 summarizes the results and discusses some open questions.

## 2    Kernels on graphs and trees

We assume that we are trying to learn a function defined on the vertices of a graph. Vertices are labelled by $i = 1 \ldots V$, instead of the generic input label $x$ we used in the introduction, and the associated function values are denoted $f_i \in \mathbb{R}$. By taking the prior $P(\boldsymbol{f})$ over these functions $\boldsymbol{f} = (f_1, \ldots, f_V)$ as a (zero mean) Gaussian process we are saying that $P(\boldsymbol{f}) \propto \exp(-\frac{1}{2}\boldsymbol{f}^{\mathrm{T}}\boldsymbol{C}^{-1}\boldsymbol{f})$. The covariance function or kernel $\boldsymbol{C}$ is then, in our graph setting, just a positive definite $V \times V$ matrix.

The graph structure is characterized by a $V \times V$ adjacency matrix, with $A_{ij} = 1$ if nodes $i$ and $j$ are connected by an edge, and 0 otherwise. All links are assumed to be undirected, so that $A_{ij} = A_{ji}$, and there are no self-loops ($A_{ii} = 0$). The degree of each node is then defined as $d_i = \sum_{j=1}^{V} A_{ij}$.

The covariance kernels we discuss in this paper are the natural generalizations of the squared-exponential kernel in Euclidean space [12]. They can be expressed in terms of the normalized graph Laplacian, defined as $\boldsymbol{L} = \boldsymbol{1} - \boldsymbol{D}^{-1/2}\boldsymbol{A}\boldsymbol{D}^{-1/2}$, where $\boldsymbol{D}$ is a diagonal matrix with entries $d_1, \ldots, d_V$ and $\boldsymbol{1}$ is the $V \times V$ identity matrix. An advantage of $\boldsymbol{L}$ over the unnormalized Laplacian $\boldsymbol{D} - \boldsymbol{A}$, which was used in the earlier paper [13], is that the eigenvalues of $\boldsymbol{L}$ (again a $V \times V$ matrix) lie in the interval [0,2] (see e.g. [14]).

From the graph Laplacian, the covariance kernels we consider here are constructed as follows. The $p$-step random walk kernel is (for $a \geq 2$)

$$\boldsymbol{C} \propto (1 - a^{-1}\boldsymbol{L})^p = \left[(1 - a^{-1})\,\boldsymbol{1} + a^{-1}\boldsymbol{D}^{-1/2}\boldsymbol{A}\boldsymbol{D}^{-1/2}\right]^p \tag{1}$$

while the diffusion kernel is given by

$$\boldsymbol{C} \propto \exp\left(-\tfrac{1}{2}\sigma^2\boldsymbol{L}\right) \propto \exp\left(\tfrac{1}{2}\sigma^2\boldsymbol{D}^{-1/2}\boldsymbol{A}\boldsymbol{D}^{-1/2}\right) \tag{2}$$

We will always normalize these so that $(1/V)\sum_i C_{ii} = 1$, which corresponds to setting the average (over vertices) prior variance of the function to be learned to unity.

To see the connection of the above kernels to random walks, assume we have a walker on the graph who at each time step selects randomly one of the neighbouring vertices and moves to it. The probability for a move from vertex $j$ to $i$ is then $A_{ij}/d_j$. The transition matrix after $s$ steps follows as $(\boldsymbol{A}\boldsymbol{D}^{-1})^s$: its $ij$-element gives the probability of being on vertex $i$, having started at $j$. We can now compare this with the $p$-step kernel by expanding the $p$-th power in (1):

$$\boldsymbol{C} \propto \sum_{s=0}^{p} \binom{p}{s} a^{-s}(1-a^{-1})^{p-s}(\boldsymbol{D}^{-1/2}\boldsymbol{A}\boldsymbol{D}^{-1/2})^s = \boldsymbol{D}^{-1/2}\sum_{s=0}^{p}\binom{p}{s}a^{-s}(1-a^{-1})^{p-s}(\boldsymbol{A}\boldsymbol{D}^{-1})^s\boldsymbol{D}^{1/2}$$

$$\tag{3}$$

Thus $\boldsymbol{C}$ is essentially a random walk transition matrix, averaged over the number of steps $s$ with

$$s \sim \mathrm{Binomial}(p, 1/a) \tag{4}$$

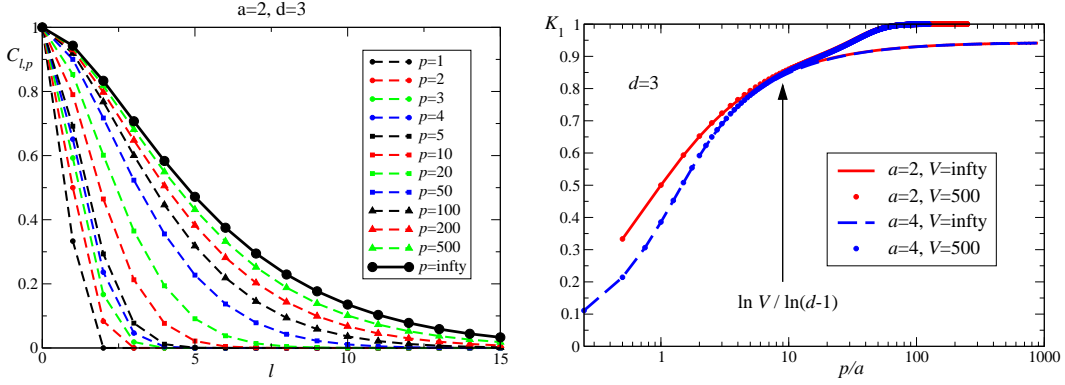

Figure 1: (Left) Random walk kernel $C_{\ell,p}$ plotted vs distance $\ell$ along graph, for increasing number of steps $p$ and $a = 2$, $d = 3$. Note the convergence to a limiting shape for large $p$ that is *not* the naive fully correlated limit $C_{\ell,p\to\infty} = 1$. (Right) Numerical results for average covariance $K_1$ between neighbouring nodes, averaged over neighbours and over randomly generated regular graphs.

This shows that $1/a$ can be interpreted as the probability of actually taking a step at each of $p$ "attempts". To obtain the actual $C$ the resulting averaged transition matrix is premultiplied by $D^{-1/2}$ and postmultiplied by $D^{1/2}$, which ensures that the kernel $C$ is symmetric. For the diffusion kernel, one finds an analogous result but the number of random walk steps is now distributed as $s \sim \mathrm{Poisson}(\sigma^2/2)$. This implies in particular that the diffusion kernel is the limit of the $p$-step kernel for $p, a \to \infty$ at constant $p/a = \sigma^2/2$. Accordingly, we discuss mainly the $p$-step kernel in this paper because results for the diffusion kernel can be retrieved as limiting cases.

In the limit of a large number of steps $s$, the random walk on a graph will reach its stationary distribution $p_\infty \propto De$ where $e = (1, \ldots, 1)$. (This form of $p_\infty$ can be verified by checking that it remains unchanged after multiplication with the transition matrix $AD^{-1}$.) The $s$-step transition matrix for large $s$ is then $p_\infty e^{\mathrm{T}} = Dee^{\mathrm{T}}$ because we converge from any starting vertex to the stationary distribution. It follows that for large $p$ or $\sigma^2$ the covariance kernel becomes $C \propto D^{1/2}ee^{\mathrm{T}}D^{1/2}$, i.e. $C_{ij} \propto (d_i d_j)^{1/2}$. This is consistent with the interpretation of $\sigma$ or $(p/a)^{1/2}$ as a lengthscale over which the random walk can diffuse along the graph: once this lengthscale becomes large, the covariance kernel $C_{ij}$ is essentially independent of the distance (along the graph) between the vertices $i$ and $j$, and the function $f$ becomes fully correlated across the graph. (Explicitly $f = vD^{1/2}e$ under the prior, with $v$ a single Gaussian random variable.) As we next show, however, the approach to this fully correlated limit as $p$ or $\sigma$ are increased is non-trivial.

We focus in this paper on kernels on random regular graphs. This means we consider adjacency matrices $A$ which are regular in the sense that they give for each vertex the same degree, $d_i = d$. A uniform probability distribution is then taken across all $A$ that obey this constraint [15]. What will the above kernels look like on typical samples drawn from this distribution? Such random regular graphs will have long loops, of length of order $\ln(V)$ or larger if $V$ is large. Their local structure is then that of a regular tree of degree $d$, which suggests that it should be possible to calculate the kernel accurately within a tree approximation. In a regular tree all nodes are equivalent, so the kernel can only depend on the distance $\ell$ between two nodes $i$ and $j$. Denoting this kernel value $C_{\ell,p}$ for a $p$-step random walk kernel, one has then $C_{\ell,p=0} = \delta_{\ell,0}$ and

$$\gamma_{p+1}C_{0,p+1} = \left(1 - \tfrac{1}{a}\right)C_{0,p} + \tfrac{1}{a}\,C_{1,p} \tag{5}$$

$$\gamma_{p+1}C_{\ell,p+1} = \tfrac{1}{ad}\,C_{\ell-1,p} + \left(1 - \tfrac{1}{a}\right)C_{\ell,p} + \tfrac{d-1}{ad}\,C_{\ell+1,p} \qquad \text{for } \ell \geq 1 \tag{6}$$

where $\gamma_p$ is chosen to achieve the desired normalization $C_{0,p} = 1$ of the prior variance for every $p$.

Fig. 1(left) shows results obtained by iterating this recursion numerically, for a regular graph (in the tree approximation) with degree $d = 3$, and $a = 2$. As expected the kernel becomes more long-ranged initially as $p$ increases, but eventually it is seen to approach a non-trivial limiting form. This can be calculated as

$$C_{\ell,p\to\infty} = [1 + \ell(d-1)/d](d-1)^{-\ell/2} \tag{7}$$

and is also plotted in the figure, showing good agreement with the numerical iteration. There are (at least) two ways of obtaining the result (7). One is to take the limit $\sigma \to \infty$ of the integral representation of the diffusion kernel on regular trees given in [16] (which is also quoted in [13] but with a typographical error that effectively removes the factor $(d-1)^{-\ell/2}$). Another route is to find the steady state of the recursion for $C_{\ell,p}$. This is easy to do but requires as input the unknown steady state value of $\gamma_p$. To determine this, one can map from $C_{\ell,p}$ to the total random walk probability $S_{\ell,p}$ in each "shell" of vertices at distance $\ell$ from the starting vertex, changing variables to $S_{0,p} = C_{0,p}$ and $S_{\ell,p} = d(d-1)^{\ell-1}C_{\ell,p}$ ($\ell \geq 1$). Omitting the factors $\gamma_p$, this results in a recursion for $S_{\ell,p}$ that simply describes a biased random walk on $\ell = 0, 1, 2, \ldots$, with a probability of $1 - 1/a$ of remaining at the current $\ell$, probability $1/(ad)$ of moving to the left and probability $(d-1)/(ad)$ of moving to the right. The point $\ell = 0$ is a reflecting barrier where only moves to the right are allowed, with probability $1/a$. The time evolution of this random walk starting from $\ell = 0$ can now be analysed as in [17]. As expected from the balance of moves to the left and right, $S_{\ell,p}$ for large $p$ is peaked around the average position of the walk, $\ell = p(d-2)/(ad)$. For $\ell$ smaller than this $S_{\ell,p}$ has a tail behaving as $\propto (d-1)^{\ell/2}$, and converting back to $C_{\ell,p}$ gives the large-$\ell$ scaling of $C_{\ell,p\to\infty} \propto (d-1)^{-\ell/2}$; this in turn fixes the value of $\gamma_{p\to\infty}$ and so eventually gives (7).

The above analysis shows that for large $p$ the random walk kernel, calculated in the absence of loops, does not approach the expected fully correlated limit; given that all vertices have the same degree, the latter would correspond to $C_{\ell,p\to\infty} = 1$. This implies, conversely, that the fully correlated limit is reached only because of the presence of loops in the graph. It is then interesting to ask at what point, as $p$ is increased, the tree approximation for the kernel breaks down. To estimate this, we note that a regular tree of depth $\ell$ has $V = 1 + d(d-1)^{\ell-1}$ nodes. So a regular graph can be tree-like at most out to $\ell \approx \ln(V)/\ln(d-1)$. Comparing with the typical number of steps our random walk takes, which is $p/a$ from (4), we then expect loop effects to appear in the covariance kernel when

$$p/a \approx \ln(V)/\ln(d-1) \tag{8}$$

To check this prediction, we measure the analogue of $C_{1,p}$ on randomly generated [15] regular graphs. Because of the presence of loops, the local kernel values are not all identical, so the appropriate estimate of what would be $C_{1,p}$ on a tree is $K_1 = C_{ij}/\sqrt{C_{ii}C_{jj}}$ for neighbouring nodes $i$ and $j$. Averaging over all pairs of such neighbours, and then over a number of randomly generated graphs we find the results in Fig. 1(right). The results for $K_1$ (symbols) accurately track the tree predictions (lines) for small $p/a$, and start to deviate just around the values of $p/a$ expected from (8), as marked by the arrow. The deviations manifest themselves in larger values of $K_1$, which eventually – now that $p/a$ is large enough for the kernel to "notice" the loops - approach the fully correlated limit $K_1 = 1$.

## 3 Learning curves

We now turn to the analysis of learning curves for GP regression on random regular graphs. We assume that the target function $\boldsymbol{f}^*$ is drawn from a GP prior with a $p$-step random walk covariance kernel $\boldsymbol{C}$. Training examples are input-output pairs $(i_\mu, f^*_{i_\mu} + \xi_\mu)$ where $\xi_\mu$ is i.i.d. Gaussian noise of variance $\sigma^2$; the distribution of training inputs $i_\mu$ is taken to be uniform across vertices. Inference from a data set $D$ of $n$ such examples $\mu = 1, \ldots, n$ takes place using the prior defined by $\boldsymbol{C}$ and a Gaussian likelihood with noise variance $\sigma^2$. We thus assume an inference model that is matched to the data generating process. This is obviously an over-simplification but is appropriate for the present first exploration of learning curves on random graphs. We emphasize that as $n$ is increased we see more and more function values from the *same* graph, which is fixed by the problem domain; the graph does not grow.

The generalization error $\epsilon$ is the squared difference between the estimated function $\hat{f}_i$ and the target $f^*_i$, averaged across the (uniform) input distribution, the posterior distribution of $\boldsymbol{f}^*$ given $D$, the distribution of datasets $D$, and finally – in our non-Euclidean setting – the random graph ensemble. Given the assumption of a matched inference model, this is just the average Bayes error, or the average posterior variance, which can be expressed explicitly as [1]

$$\epsilon(n) = V^{-1} \sum_i \left\langle C_{ii} - \boldsymbol{k}(i)^{\mathrm{T}} \boldsymbol{K} \boldsymbol{k}^{-1}(i) \right\rangle_{D,\mathrm{graphs}} \tag{9}$$

where the average is over data sets and over graphs, $\boldsymbol{K}$ is an $n \times n$ matrix with elements $K_{\mu\mu'} = C_{i_\mu, i_{\mu'}} + \sigma^2 \delta_{\mu\mu'}$ and $\boldsymbol{k}(i)$ is a vector with entries $k_\mu(i) = C_{i,i_\mu}$. The resulting learning curve depends, in addition to $n$, on the graph structure as determined by $V$ and $d$, and the kernel and noise level as specified by $p$, $a$ and $\sigma^2$. We fix $d = 3$ throughout to avoid having too many parameters to vary, although similar results are obtained for larger $d$.

Exact prediction of learning curves by analytical calculation is very difficult due to the complicated way in which the random selection of training inputs enters the matrix $\boldsymbol{K}$ and vector $\boldsymbol{k}$ in (9). However, by first expressing these quantities in terms of kernel eigenvalues (see below) and then approximating the average over datasets, one can derive the approximation [3, 6]

$$\epsilon = g\left(\frac{n}{\epsilon + \sigma^2}\right), \qquad g(h) = \sum_{\alpha=1}^{V}(\lambda_\alpha^{-1} + h)^{-1} \tag{10}$$

This equation for $\epsilon$ has to be solved self-consistently because $\epsilon$ also appears on the r.h.s. In the Euclidean case the resulting predictions approximate the true learning curves quite reliably. The derivation of (10) for inputs on a fixed graph is unchanged from [3], provided the kernel eigenvalues $\lambda_\alpha$ appearing in the function $g(h)$ are defined appropriately, by the eigenfunction condition $\langle C_{ij}\phi_j \rangle = \lambda\phi_i$; the average here is over the input distribution, i.e. $\langle \ldots \rangle = V^{-1}\sum_j \ldots$ From the definition (1) of the $p$-step kernel, we see that then $\lambda_\alpha = \kappa V^{-1}(1 - \lambda_\alpha^L/a)^p$ in terms of the corresponding eigenvalue of the graph Laplacian $L$. The constant $\kappa$ has to be chosen to enforce our normalization convention $\sum_\alpha \lambda_\alpha = \langle C_{jj} \rangle = 1$.

Fortunately, for large $V$ the spectrum of the Laplacian of a random regular graph can be approximated by that of the corresponding large regular tree, which has spectral density [14]

$$\rho(\lambda^L) = \frac{\sqrt{\frac{4(d-1)}{d^2} - (\lambda^L - 1)^2}}{2\pi d\lambda^L(2 - \lambda^L)} \tag{11}$$

in the range $\lambda^L \in [\lambda_-^L, \lambda_+^L]$, $\lambda_\pm^L = 1 + 2d^{-1}(d-1)^{1/2}$, where the term under the square root is positive. (There are also two isolated eigenvalues $\lambda^L = 0, 2$ but these have weight $1/V$ each and so can be ignored for large $V$.) Rewriting (10) as $\epsilon = V^{-1}\sum_\alpha[(V\lambda_\alpha)^{-1} + (n/V)(\epsilon + \sigma^2)^{-1}]^{-1}$ and then replacing the average over kernel eigenvalues by an integral over the spectral density leads to the following prediction for the learning curve:

$$\epsilon = \int d\lambda^L \rho(\lambda^L)[\kappa^{-1}(1 - \lambda^L/a)^{-p} + \nu/(\epsilon + \sigma^2)]^{-1} \tag{12}$$

with $\kappa$ determined from $\kappa \int d\lambda^L \rho(\lambda^L)(1 - \lambda^L/a)^p = 1$. A general consequence of the form of this result is that the learning curve depends on $n$ and $V$ only through the ratio $\nu = n/V$, i.e. the number of training examples per vertex. The approximation (12) also predicts that the learning curve will have two regimes, one for small $\nu$ where $\epsilon \gg \sigma^2$ and the generalization error will be essentially independent of $\sigma^2$; and another for large $\nu$ where $\epsilon \ll \sigma^2$ so that $\epsilon$ can be neglected on the r.h.s. and one has a fully explicit expression for $\epsilon$.

We compare the above prediction in Fig. 2(left) to the results of numerical simulations of the learning curves, averaged over datasets and random regular graphs. The two regimes predicted by the approximation are clearly visible; the approximation works well inside each regime but less well in the crossover between the two. One striking observation is that the approximation seems to predict the asymptotic large-$n$ behaviour exactly; this is distinct to the Euclidean case, where generally only the power-law of the $n$-dependence but not its prefactor come out accurately. To see why, we exploit that for large $n$ (where $\epsilon \ll \sigma^2$) the approximation (9) effectively neglects fluctuations in the training input "density" of a randomly drawn set of training inputs [3, 6]. This is justified in the graph case for large $\nu = n/V$, because the number of training inputs each vertex receives, Binomial$(n, 1/V)$, has negligible relative fluctuations away from its mean $\nu$. In the Euclidean case there is no similar result, because all training inputs are different with probability one even for large $n$.

Fig. 2(right) illustrates that for larger $a$ the difference in the crossover region between the true (numerically simulated) learning curves and our approximation becomes larger. This is because the average number of steps $p/a$ of the random walk kernel then decreases: we get closer to the limit of uncorrelated function values ($a \to \infty$, $C_{ij} = \delta_{ij}$). In that limit and for low $\sigma^2$ and large $V$ the

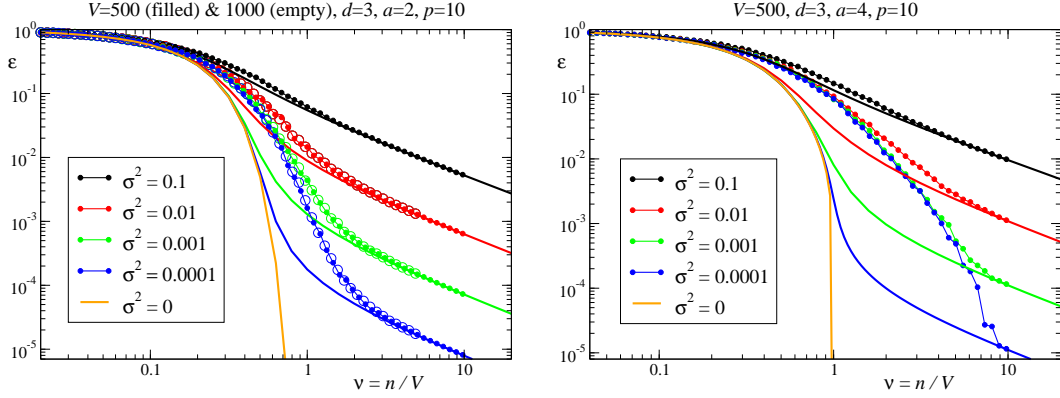

Figure 2: (Left) Learning curves for GP regression on random regular graphs with degree $d = 3$ and $V = 500$ (small filled circles) and $V = 1000$ (empty circles) vertices. Plotting generalization error versus $\nu = n/V$ superimposes the results for both values of $V$, as expected from the approximation (12). The lines are the quantitative predictions of this approximation. Noise level as shown, kernel parameters $a = 2$, $p = 10$. (Right) As on the left but with $V = 500$ only and for larger $a = 4$.

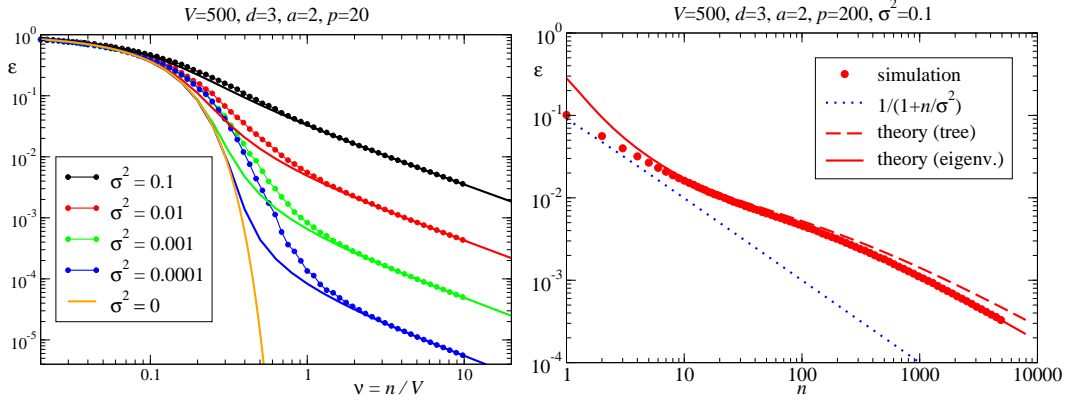

Figure 3: (Left) Learning curves for GP regression on random regular graphs with degree $d = 3$ and $V = 500$, and kernel parameters $a = 2$, $p = 20$; noise level $\sigma^2$ as shown. Circles: numerical simulations; lines: approximation (12). (Right) As on the left but for much larger $p = 200$ and for a single random graph, with $\sigma^2 = 0.1$. Dotted line: naive estimate $\epsilon = 1/(1 + n/\sigma^2)$. Dashed line: approximation (10) using the tree spectrum and the large $p$-limit, see (17). Solid line: (10) with numerically determined graph eigenvalues $\lambda_\alpha^L$ as input.

true learning curve is $\epsilon = \exp(-\nu)$, reflecting the probability of a training input set not containing a particular vertex, while the approximation can be shown to predict $\epsilon = \max\{1 - \nu, 0\}$, i.e. a decay of the error to zero at $\nu = 1$. Plotting these two curves (not displayed here) indeed shows the same "shape" of disagreement as in Fig. 2(right), with the approximation underestimating the true generalization error.

Increasing $p$ has the effect of making the kernel longer ranged, giving an effect opposite to that of increasing $a$. In line with this, larger values of $p$ improve the accuracy of the approximation (12): see Fig. 3(left).

One may ask about the shape of the learning curves for large number of training examples (per vertex) $\nu$. The roughly straight lines on the right of the log-log plots discussed so far suggest that $\epsilon \propto 1/\nu$ in this regime. This is correct in the mathematical limit $\nu \to \infty$ because the graph kernel has a nonzero minimal eigenvalue $\lambda_- = \kappa V^{-1}(1 - \lambda_+^L/a)^p$: for $\nu \gg \sigma^2/(V\lambda_-)$, the square bracket

in (12) can then be approximated by $\nu/(\epsilon+\sigma^2)$ and one gets (because also $\epsilon \ll \sigma^2$ in the asymptotic regime) $\epsilon \approx \sigma^2/\nu$.

However, once $p$ becomes reasonably large, $V\lambda_-$ can be shown – by analysing the scaling of $\kappa$, see Appendix – to be extremely (exponentially in $p$) small; for the parameter values in Fig. 3(left) it is around $4 \times 10^{-30}$. The "terminal" asymptotic regime $\epsilon \approx \sigma^2/\nu$ is then essentially unreachable. A more detailed analysis of (12) for large $p$ and large (but not exponentially large) $\nu$, as sketched in the Appendix, yields

$$\epsilon \propto (c\sigma^2/\nu)\ln^{3/2}(\nu/(c\sigma^2)), \qquad c \propto p^{-3/2} \tag{13}$$

This shows that there are logarithmic corrections to the naive $\sigma^2/\nu$ scaling that would apply in the true terminal regime. More intriguing is the scaling of the coefficient $c$ with $p$, which implies that to reach a specified (low) generalization error one needs a number of training examples per vertex of order $\nu \propto c\sigma^2 \propto p^{-3/2}\sigma^2$. Even though the covariance kernel $C_{\ell,p}$ – in the same tree approximation that also went into (12) – approaches a limiting form for large $p$ as discussed in Sec. 2, generalization performance thus continues to improve with increasing $p$. The explanation for this must presumably be that $C_{\ell,p}$ converges to the limit (7) only at fixed $\ell$, while in the tail $\ell \propto p$, it continues to change.

For finite graph sizes $V$ we know of course that loops will eventually become important as $p$ increases, around the crossover point estimated in (8). The approximation for the learning curve in (12) should then break down. The most naive estimate beyond this point would be to say that the kernel becomes nearly fully correlated, $C_{ij} \propto (d_i d_j)^{1/2}$ which in the regular case simplifies to $C_{ij} = 1$. With only one function value to learn, and correspondingly only one nonzero kernel eigenvalue $\lambda_{\alpha=1} = 1$, one would predict $\epsilon = 1/(1 + n/\sigma^2)$. Fig. 3(right) shows, however, that this significantly underestimates the actual generalization error, even though for this graph $\lambda_{\alpha=1} = 0.994$ is very close to unity so that the other eigenvalues sum to no more than 0.006. An almost perfect prediction is obtained, on the other hand, from the approximation (10) with the numerically calculated values of the Laplacian – and hence kernel – eigenvalues. The presence of the small kernel eigenvalues is again seen to cause logarithmic corrections to the naive $\epsilon \propto 1/n$ scaling. Using the tree spectrum as an approximation and exploiting the large-$p$ limit, one finds indeed (see Appendix, Eq. (17)) that $\epsilon \propto (c'\sigma^2/n)\ln^{3/2}(n/c'\sigma^2)$ where now $n$ enters rather than $\nu = n/V$, $c'$ being a constant dependent only on $p$ and $a$: informally, the function to be learned only has a finite (rather than $\propto V$) number of degrees of freedom. The approximation (17) in fact provides a qualitatively accurate description of the data Fig. 3(right), as the dashed line in the figure shows. We thus have the somewhat unusual situation that the tree spectrum is enough to give a good description of the learning curves even when loops are important, while (see Sec. 2) this is not so as far as the evaluation of the covariance kernel itself is concerned.

## 4   Summary and Outlook

We have studied theoretically the generalization performance of GP regression on graphs, focussing on the paradigmatic case of random regular graphs where every vertex has the same degree $d$. Our initial concern was with the behaviour of $p$-step random walk kernels on such graphs. If these are calculated within the usual approximation of a locally tree-like structure, then they converge to a non-trivial limiting form (7) when $p$ – or the corresponding lengthscale $\sigma$ in the closely related diffusion kernel – becomes large. The limit of full correlation between all function values on the graph is only reached because of the presence of loops, and we have estimated in (8) the values of $p$ around which the crossover to this loop-dominated regime occurs; numerical data for correlations of function values on neighbouring vertices support this result.

In the second part of the paper we concentrated on the learning curves themselves. We assumed that inference is performed with the correct parameters describing the data generating process; the generalization error is then just the Bayes error. The approximation (12) gives a good qualitative description of the learning curve using only the known spectrum of a large regular tree as input. It predicts in particular that the key parameter that determines the generalization error is $\nu = n/V$, the number of training examples per vertex. We demonstrated also that the approximation is in fact more useful than in the Euclidean case because it gives exact asymptotics for the limit $\nu \gg 1$.

Quantitatively, we found that the learning curves decay as $\epsilon \propto \sigma^2/\nu$ with non-trivial logarithmic correction terms. Slower power laws $\propto \nu^{-\alpha}$ with $\alpha < 1$, as in the Euclidean case, do not appear.

We attribute this to the fact that on a graph there is no analogue of the local roughness of a target function because there is a minimum distance (one step along the graph) between different input points. Finally we looked at the learning curves for larger $p$, where loops become important. These can still be predicted quite accurately by using the tree eigenvalue spectrum as an approximation, if one keeps track of the zero graph Laplacian eigenvalue which we were able to ignore previously; the approximation shows that the generalization error scales as $\sigma^2/n$ with again logarithmic corrections.

In future work we plan to extend our analysis to graphs that are not regular, including ones from application domains as well as artificial ones with power-law tails in the distribution of degrees $d$, where qualitatively new effects are to be expected. It would also be desirable to improve the predictions for the learning curve in the crossover region $\epsilon \approx \sigma^2$, which should be achievable using iterative approaches based on belief propagation that have already been shown to give accurate approximations for graph eigenvalue spectra [18]. These tools could then be further extended to study e.g. the effects of model mismatch in GP regression on random graphs, and how these are mitigated by tuning appropriate hyperparameters.

## Appendix

We sketch here how to derive (13) from (12) for large $p$. Eq. (12) writes $\epsilon = g(\nu V/(\epsilon + \sigma^2))$ with

$$g(h) = \int_{\lambda_-^L}^{\lambda_+^L} d\lambda^L \, \rho(\lambda^L)[\kappa^{-1}(1 - \lambda^L/a)^{-p} + hV^{-1}]^{-1} \tag{14}$$

and $\kappa$ determined from the condition $g(0) = 1$. (This $g(h)$ is the tree spectrum approximation to the $g(h)$ of (10).) Turning first to $g(0)$, the factor $(1 - \lambda^L/a)^p$ decays quickly to zero as $\lambda^L$ increases above $\lambda_-^L$. One can then approximate this factor according to $(1 - \lambda_-^L/a)^p[(a - \lambda^L)/(a - \lambda_-^L)]^p \approx (1 - \lambda_-^L/a)^p \exp[-(\lambda^L - \lambda_-^L)p/(a - \lambda_-^L)]$. In the regime near $\lambda_-^L$ one can also approximate the spectral density (11) by its leading square-root increase, $\rho(\lambda^L) = r(\lambda^L - \lambda_-^L)^{1/2}$, with $r = (d - 1)^{1/4}d^{5/2}/[\pi(d-2)^2]$. Switching then to a new integration variable $y = (\lambda^L - \lambda_-^L)p/(a - \lambda_-^L)$ and extending the integration limit to $\infty$ gives

$$1 = g(0) = \kappa r(1 - \lambda_-^L/a)^p[p/(a - \lambda_-^L)]^{-3/2} \int_0^\infty dy \sqrt{y} \, e^{-y} \tag{15}$$

and this fixes $\kappa$. Proceeding similarly for $h > 0$ gives

$$g(h) = \kappa r(1 - \lambda_-^L/a)^p[p/(a - \lambda_-^L)]^{-3/2} F(h\kappa V^{-1}(1 - \lambda_-^L/a)^p), \qquad F(z) = \int_0^\infty dy \sqrt{y} \, (e^y + z)^{-1} \tag{16}$$

Dividing by $g(0) = 1$ shows that simply $g(h) = F(hV^{-1}c^{-1})/F(0)$, where $c = 1/[\kappa(1 - \lambda_-^L/a)^p] = rF(0)[p/(a - \lambda_-^L)]^{-3/2}$ which scales as $p^{-3/2}$. In the asymptotic regime $\epsilon \ll \sigma^2$ we then have $\epsilon = g(\nu V/\sigma^2) = F(\nu/(c\sigma^2))/F(0)$ and the desired result (13) follows from the large-$z$ behaviour of $F(z) \approx z^{-1} \ln^{3/2}(z)$.

One can proceed similarly for the regime where loops become important. Clearly the zero Laplacian eigenvalue with weight $1/V$ then has to be taken into account. If we assume that the remainder of the Laplacian spectrum can still be approximated by that of a tree [18], we get

$$g(h) = \frac{(V + h\kappa)^{-1} + r(1 - \lambda_-^L/a)^p[p/(a - \lambda_-^L)]^{-3/2}F(h\kappa V^{-1}(1 - \lambda_-^L/a)^p)}{V^{-1} + r(1 - \lambda_-^L/a)^p[p/(a - \lambda_-^L)]^{-3/2}F(0)} \tag{17}$$

The denominator here is $\kappa^{-1}$ and the two terms are proportional respectively to the covariance kernel eigenvalue $\lambda_1$, corresponding to $\lambda_1^L = 0$ and the constant eigenfunction, and to $1 - \lambda_1$. Dropping the first terms in the numerator and denominator of (17) by taking $V \to \infty$ leads back to the previous analysis as it should. For a situation as in Fig. 3(right), on the other hand, where $\lambda_1$ is close to unity, we have $\kappa \approx V$ and so

$$g(h) \approx (1 + h)^{-1} + rV(1 - \lambda_-^L/a)^p[p/(a - \lambda_-^L)]^{-3/2}F(h(1 - \lambda_-^L/a)^p) \tag{18}$$

The second term, coming from the small kernel eigenvalues, is the more slowly decaying because it corresponds to fine detail of the target function that needs many training examples to learn accurately. It will therefore dominate the asymptotic behaviour of the learning curve: $\epsilon = g(n/\sigma^2) \propto F(n/(c'\sigma^2))$ with $c' = (1 - \lambda_-^L/a)^{-p}$ independent of $V$. The large-$n$ tail of the learning curve in Fig. 3(right) is consistent with this form.

# References

[1] C E Rasmussen and C K I Williams. Gaussian processes for regression. In D S Touretzky, M C Mozer, and M E Hasselmo, editors, *Advances in Neural Information Processing Systems 8*, pages 514–520, Cambridge, MA, 1996. MIT Press.

[2] M Opper. Regression with Gaussian processes: Average case performance. In I K Kwok-Yee, M Wong, I King, and Dit-Yun Yeung, editors, *Theoretical Aspects of Neural Computation: A Multidisciplinary Perspective*, pages 17–23. Springer, 1997.

[3] P Sollich. Learning curves for Gaussian processes. In M S Kearns, S A Solla, and D A Cohn, editors, *Advances in Neural Information Processing Systems 11*, pages 344–350, Cambridge, MA, 1999. MIT Press.

[4] M Opper and F Vivarelli. General bounds on Bayes errors for regression with Gaussian processes. In M Kearns, S A Solla, and D Cohn, editors, *Advances in Neural Information Processing Systems 11*, pages 302–308, Cambridge, MA, 1999. MIT Press.

[5] C K I Williams and F Vivarelli. Upper and lower bounds on the learning curve for Gaussian processes. *Mach. Learn.*, 40(1):77–102, 2000.

[6] D Malzahn and M Opper. Learning curves for Gaussian processes regression: A framework for good approximations. In T K Leen, T G Dietterich, and V Tresp, editors, *Advances in Neural Information Processing Systems 13*, pages 273–279, Cambridge, MA, 2001. MIT Press.

[7] D Malzahn and M Opper. A variational approach to learning curves. In T G Dietterich, S Becker, and Z Ghahramani, editors, *Advances in Neural Information Processing Systems 14*, pages 463–469, Cambridge, MA, 2002. MIT Press.

[8] P Sollich and A Halees. Learning curves for Gaussian process regression: approximations and bounds. *Neural Comput.*, 14(6):1393–1428, 2002.

[9] P Sollich. Gaussian process regression with mismatched models. In T G Dietterich, S Becker, and Z Ghahramani, editors, *Advances in Neural Information Processing Systems 14*, pages 519–526, Cambridge, MA, 2002. MIT Press.

[10] P Sollich. Can Gaussian process regression be made robust against model mismatch? In *Deterministic and Statistical Methods in Machine Learning*, volume 3635 of *Lecture Notes in Artificial Intelligence*, pages 199–210. 2005.

[11] M Herbster, M Pontil, and L Wainer. Online learning over graphs. In *ICML '05: Proceedings of the 22nd international conference on Machine learning*, pages 305–312, New York, NY, USA, 2005. ACM.

[12] A J Smola and R Kondor. Kernels and regularization on graphs. In M Warmuth and B Schölkopf, editors, *Proc. Conference on Learning Theory (COLT)*, Lect. Notes Comp. Sci., pages 144–158. Springer, Heidelberg, 2003.

[13] R I Kondor and J D Lafferty. Diffusion kernels on graphs and other discrete input spaces. In *ICML '02: Proceedings of the Nineteenth International Conference on Machine Learning*, pages 315–322, San Francisco, CA, USA, 2002. Morgan Kaufmann.

[14] F R K Chung. *Spectral graph theory*. Number 92 in Regional Conference Series in Mathematics. Americal Mathematical Society, 1997.

[15] A Steger and N C Wormald. Generating random regular graphs quickly. *Combinator. Probab. Comput.*, 8(4):377–396, 1999.

[16] F Chung and S-T Yau. Coverings, heat kernels and spanning trees. *The Electronic Journal of Combinatorics*, 6(1):R12, 1999.

[17] C Monthus and C Texier. Random walk on the Bethe lattice and hyperbolic brownian motion. *J. Phys. A*, 29(10):2399–2409, 1996.

[18] T Rogers, I Perez Castillo, R Kuehn, and K Takeda. Cavity approach to the spectral density of sparse symmetric random matrices. *Phys. Rev. E*, 78(3):031116, 2008.

